# New Outer Bounds on the Marginal Polytope

**David Sontag**    **Tommi Jaakkola**
Computer Science and Artificial Intelligence Laboratory
Massachusetts Institute of Technology
Cambridge, MA 02139
`dsontag,tommi@csail.mit.edu`

## Abstract

We give a new class of outer bounds on the marginal polytope, and propose a cutting-plane algorithm for efficiently optimizing over these constraints. When combined with a concave upper bound on the entropy, this gives a new variational inference algorithm for probabilistic inference in discrete Markov Random Fields (MRFs). Valid constraints on the marginal polytope are derived through a series of projections onto the cut polytope. As a result, we obtain tighter upper bounds on the log-partition function. We also show empirically that the approximations of the marginals are significantly more accurate when using the tighter outer bounds. Finally, we demonstrate the advantage of the new constraints for finding the MAP assignment in protein structure prediction.

## 1 Introduction

Graphical models such as *Markov Random Fields* (MRFs) have been successfully applied to a wide variety of fields, from computer vision to computational biology. From the point of view of inference, we are generally interested in two questions: finding the marginal probabilities of specific subsets of the variables, and finding the *Maximum a Posteriori* (MAP) assignment. Both of these require approximate methods.

We focus on a particular class of variational approximation methods that cast the inference problem as a non-linear optimization over the *marginal polytope*, the set of valid marginal probabilities. The selection of appropriate marginals from the marginal polytope is guided by the (non-linear) entropy function. Both the marginal polytope and the entropy are difficult to characterize in general, reflecting the hardness of exact inference calculations. Most message-passing algorithms for evaluating marginals, including belief propagation and tree-reweighted sum-product (TRW), operate instead within the *local consistency polytope*, characterized by pairwise consistent marginals. For general graphs, this is an outer bound of the marginal polytope. Various approximations have also been suggested for the entropy function. For example, in the TRW algorithm [10], the entropy is decomposed into a weighted combination of entropies of tree-structured distributions.

Our goal here is to provide tighter outer bounds on the marginal polytope. We show how this can be achieved efficiently using a *cutting-plane algorithm*, iterating between solving a relaxed problem and adding additional constraints. Cutting-plane algorithms are a well-known technique for solving integer linear programs. The key to such approaches is to have an efficient separation algorithm which, given an infeasible solution, can quickly find a violated constraint, generally from a very large class of valid constraints on the set of integral solutions.

The motivation for our approach comes from the cutting-plane literature for the maximum cut problem. Barahona et al. [3] showed that the MAP problem in pairwise *binary* MRFs is equivalent to a linear optimization over the cut polytope, which is the convex hull of all valid graph cuts. Tighter relaxations were obtained by using a separation algorithm together with the cutting-plane methodology. We extend this work by deriving a new class of outer bounds on the marginal polytope for

*non-binary* and *non-pairwise* MRFs. The key realization is that valid constraints can be constructed by a series of *projections* onto the cut polytope[1]. More broadly, we seek to highlight emerging connections between polyhedral combinatorics and probabilistic inference.

## 2 Background

**Markov Random Fields.** Let $\mathbf{x} \in \chi^n$ denote a random vector on $n$ variables, where, for simplicity, each variable $x_i$ takes on the values in $\chi_i = \{0, 1, \ldots, k-1\}$. The MRF is specified by a set of $d$ real valued *potentials* or *sufficient statistics* $\phi(\mathbf{x}) = \{\phi_k(\mathbf{x})\}$ and a parameter vector $\theta \in \mathbb{R}^d$:

$$p(\mathbf{x}; \theta) = \exp\{\langle \theta, \phi(\mathbf{x})\rangle - A(\theta)\}, \quad A(\theta) = \log \textstyle\sum_{\mathbf{x} \in \chi^n} \exp\{\langle \theta, \phi(\mathbf{x})\rangle\}$$

where $\langle \theta, \phi(\mathbf{x})\rangle$ denotes the dot product of the parameters and the sufficient statistics. In *pairwise MRFs*, potentials are restricted to be at most over the edges in the graph. We assume that the potentials are indicator functions, i.e., $\phi_{i,s}(\mathbf{x}) = \delta(x_i = s)$, and make use of the following notation: $\mu_{i;s} = E_\theta[\phi_{i;s}(\mathbf{x})] = p(x_i = s; \theta)$ and $\mu_{ij;st} = E_\theta[\phi_{ij;st}(\mathbf{x})] = p(x_i = s, x_j = t; \theta)$.

**Variational inference.** The inference task is to evaluate the mean vector $\mu = E_\theta[\phi(\mathbf{x})]$. The log-partition function $A(\theta)$, a convex function of $\theta$, plays a critical role in these calculations. In particular, we can write the log-partition function in terms of its Fenchel-Legendre conjugate [11]:

$$A(\theta) = \sup_{\mu \in \mathcal{M}} \{\langle \theta, \mu\rangle - B(\mu)\}, \tag{1}$$

where $B(\mu) = -H(\mu)$ is the negative entropy of the distribution parameterized by $\mu$ and is also convex. $\mathcal{M}$ is the set of realizable mean vectors $\mu$ known as the *marginal polytope*. More precisely, $\mathcal{M} := \{\mu \in \mathbb{R}^d \mid \exists p(\mathbf{x}) \text{ s.t. } \mu = E_p[\phi(\mathbf{x})]\}$. The value $\mu^* \in \mathcal{M}$ that maximizes (1) is precisely the desired mean vector corresponding to $p(\mathbf{x}; \theta)$.

Both $\mathcal{M}$ and the entropy $H(\mu)$ are difficult to characterize in general and have to be approximated. We call the resulting approximate mean vectors *pseudomarginals*. Mean field algorithms optimize over an *inner bound* on the marginal polytope (which is not convex) by restricting the marginal vectors to those coming from simpler, e.g., fully factored, distributions. The entropy can be evaluated exactly in this case (the distribution is simple). Alternatively, we can relax the optimization to be over an *outer bound* on the marginal polytope and also bound the entropy function.

Most message passing algorithms for evaluating marginal probabilities obtain locally consistent beliefs so that the pseudomarginals over the edges agree with the singleton pseudomarginals at the nodes. The solution is therefore sought within the *local marginal polytope*

$$\text{LOCAL}(G) = \{\mu \geq 0 \mid \textstyle\sum_{s \in \chi_i} \mu_{i;s} = 1, \sum_{t \in \chi_j} \mu_{ij;st} = \mu_{i;s}\} \tag{2}$$

Clearly, $\mathcal{M} \subseteq \text{LOCAL}(G)$ since true marginals are also locally consistent. For trees, $\mathcal{M} = \text{LOCAL}(G)$. Both LOCAL(G) and $\mathcal{M}$ have the same integral vertices for general graphs [11, 6].

Belief propagation can be seen as optimizing pseudomarginals over $\text{LOCAL}(G)$ with a (non-convex) *Bethe approximation* to the entropy [15]. The tree-reweighted sum-product algorithm [10], on the other hand, uses a concave upper bound on the entropy, expressed as a convex combination of entropies corresponding to the spanning trees of the original graph. The log-determinant relaxation [12] is instead based on a semi-definite outer bound on the marginal polytope combined with a Gaussian approximation to the entropy function. Since the moment matrix $M_1(\mu)$ can be written as $E_\theta[(1\ \mathbf{x})^T(1\ \mathbf{x})]$ for $\mu \in \mathcal{M}$, the outer bound is obtained simply by requiring only that the pseudomarginals lie in $\text{SDEF}_1(K_n) = \{\mu \in \mathbb{R}^+ \mid M_1(\mu) \succeq 0\}$.

**Maximum a posteriori.** The marginal polytope also plays a critical role in finding the MAP assignment. The problem is to find an assignment $\mathbf{x} \in \chi^n$ which maximizes $p(\mathbf{x}; \theta)$, or equivalently:

$$\max_{\mathbf{x} \in \chi^n} \log p(\mathbf{x}; \theta) = \max_{\mathbf{x} \in \chi^n} \langle \theta, \phi(\mathbf{x})\rangle - A(\theta) = \sup_{\mu \in \mathcal{M}} \langle \theta, \mu\rangle - A(\theta) \tag{3}$$

where the log-partition function $A(\theta)$ remains a constant and can be ignored. The last equality holds because the optimal value of the linear program is obtained at a vertex (integral solution). That is, when the MAP assignment $\mathbf{x}^*$ is unique, the maximizing $\mu^*$ is $\phi(\mathbf{x}^*)$.

**Algorithm 1** Cutting-plane algorithm for probabilistic inference

---
1: OUTER ← LOCAL(G)
2: **repeat**
3:     $\mu^* \leftarrow \text{argmax}_{\mu \in \text{OUTER}} \{\langle \theta, \mu \rangle - B^*(\mu)\}$
4:     Choose projection graph $G_\pi$, e.g. *single*, $k$, or *full*
5:     $\mathcal{C} \leftarrow \texttt{Find\_Violated\_Inequalities}(G_\pi, \Psi_\pi(\mu^*))$
6:     OUTER ← OUTER $\cap$ $\mathcal{C}$
7: **until** $\mathcal{C} = \mathbb{R}^d$ (did not find any violated inequalities)

---

**Cycle inequalities.** The marginal polytope can be defined by the intersection of a large number of linear inequalities. We focus on inequalities beyond those specifying LOCAL($G$), in particular the *cycle inequalities* [4, 2, 6]. Assume the variables are binary. Given an assignment $\mathbf{x} \in \{0,1\}^n$, $(i,j) \in E$ is *cut* if $x_i \neq x_j$. The cycle inequalities arise from the observation that a cycle must have an even (possibly zero) number of cut edges. Suppose we start at node $i$, where $x_i = 0$. As we traverse the cycle, the assignment changes each time we cross a cut edge. Since we must return to $x_i = 0$, the assignment can only change an even number of times. For a cycle $C$ and any $F \subseteq C$ such that $|F|$ is odd, this constraint can be written as $\sum_{(i,j) \in C \setminus F} \delta(x_i \neq x_j) + \sum_{(i,j) \in F} \delta(x_i = x_j) \geq 1$. Since this constraint is valid for all assignments $\mathbf{x} \in \{0,1\}^n$, it holds also in expectation. Thus

$$\sum_{(i,j) \in C \setminus F} (\mu_{ij;10} + \mu_{ij;01}) + \sum_{(i,j) \in F} (\mu_{ij;00} + \mu_{ij;11}) \geq 1 \qquad (4)$$

is valid for any $\mu \in \mathcal{M}_{\{0,1\}}$, the marginal polytope of a binary pairwise MRF. For a chordless circuit $C$, the cycle inequalities are facets of $\mathcal{M}_{\{0,1\}}$ [4]. They suffice to characterize $\mathcal{M}_{\{0,1\}}$ for a graph $G$ if and only if $G$ has no $K_4$-minor. Although there are exponentially many cycles and cycle inequalities for a graph, Barahona and Mahjoub [4, 6] give a simple algorithm to separate the whole class of cycle inequalities.

To see whether any cycle inequality is violated, construct the undirected graph $G' = (V', E')$ where $V'$ contains nodes $i_1$ and $i_2$ for each $i \in V$, and for each $(i,j) \in E$, the edges in $E'$ are: $(i_1, j_1)$ and $(i_2, j_2)$ with weight $\mu_{ij;10} + \mu_{ij;01}$, and $(i_1, j_2)$ and $(i_2, j_1)$ with weight $\mu_{ij;00} + \mu_{ij;11}$. Then, for each node $i \in V$ we find the shortest path in $G'$ from $i_1$ to $i_2$. The shortest of all these paths will not use both copies of any node $j$ (otherwise the path $j_1$ to $j_2$ would be shorter), and so defines a cycle in $G$ and gives the minimum value of $\sum_{(i,j) \in C \setminus F} (\mu_{ij;10} + \mu_{ij;01}) + \sum_{(i,j) \in F} (\mu_{ij;00} + \mu_{ij;11})$. If this is less than 1, we have found a violated cycle inequality; otherwise, $\mu$ satisfies all cycle inequalities. Using Dijkstra's shortest paths algorithm with a Fibonacci heap [5], the separation problem can be solved in time $O(n^2 \log n + n|E|)$.

## 3 Cutting-plane algorithm

Our main result is the proposed Algorithm 1 given above. The algorithm alternates between solving for an upper bound of the log-partition function (see Eq. 1) and tightening the outer bound on the marginal polytope by incorporating valid constraints that are violated by the current pseudo-marginals. The projection graph (line 4) is not needed for binary pairwise MRFs and will be described in the next section. We start the algorithm (line 1) with the loose outer bound on the marginal polytope given by the local consistency constraints. Tighter initial constraints, e.g., $M_1(\mu) \succeq 0$, are possible as well.

The separation algorithm returns a feasible set $\mathcal{C}$ given by the intersection of halfspaces, and we intersect this with OUTER to obtain a smaller feasible space, i.e. a tighter relaxation. The experiments in Section 5 use the separation algorithm for cycle inequalities. However, any class of valid constraints for the marginal polytope with an efficient separation algorithm may be used in line 5. Other examples besides the cycle inequalities include the odd-wheel and bicycle odd-wheel inequalities [6], and also linear inequalities that enforce positive semi-definiteness of $M_1(\mu)$. The cutting-plane algorithm is in effect optimizing the variational objective (Eq. 1) over a relaxation of the marginal polytope defined by the intersection of all inequalities that can be returned in line 5.

Any entropy approximation $B^*(\mu)$ can be used so long as we can efficiently solve the optimization problem in line 3. The log-determinant and TRW entropy approximations have two appealing fea-

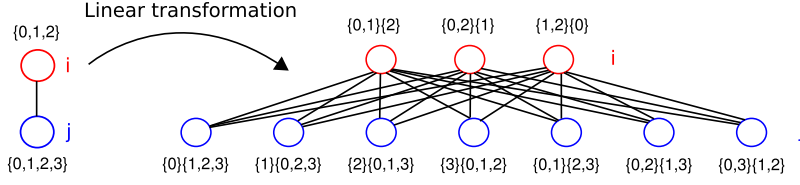

Figure 1: Illustration of the projection $\Psi_\pi$ for one edge $(i,j) \in E$ where $\chi_i = \{0,1,2\}$ and $\chi_j = \{0,1,2,3\}$. The projection graph $G_\pi$, shown on the right, has 3 partitions for $i$ and 7 for $j$.

tures. First, as upper bounds they permit the algorithm to be used for obtaining tighter upper bounds on the log-partition function. Second, the objective functions to be maximized are convex and can be solved efficiently using conditional gradient or other methods.

When the algorithm terminates, we can use the last $\mu^*$ vector as an approximation to the single node and edge marginals. The results given in Section 5 use this method. The algorithm for MAP is the same, excluding the entropy function in line 3; the optimization is simply a linear program. Since all integral vectors in the relaxation OUTER are extreme points of the marginal polytope, any integral $\mu^*$ is the MAP assignment.

## 4    Generalization to non-binary MRFs

In this section we give a new class of valid inequalities for the marginal polytope of non-binary and non-pairwise MRFs, and show how to efficiently separate this exponentially large set of inequalities. The key theoretical idea is to project the marginal polytope onto different binary marginal polytopes. Aggregation and projection are well-known techniques in polyhedral combinatorics for obtaining valid inequalities [6]. Given a linear projection $\Phi(\mathbf{x}) = A\mathbf{x}$, any valid inequality $\mathbf{c}'\Phi(\mathbf{x}) \leq \mathbf{b}$ for $\Phi(\mathbf{x})$ also gives the valid inequality $\mathbf{c}'A\mathbf{x} \leq \mathbf{b}$ for $\mathbf{x}$. We obtain new inequalities by aggregating the *values* of each variable.

For each variable $i$, let $\pi_i^q$ be a *partition* of its values into two non-empty sets, i.e., the map $\pi_i^q : \chi_i \to \{0,1\}$ is surjective. Let $\pi_\mathbf{i} = \{\pi_i^1, \pi_i^2, \ldots\}$ be a *collection of partitions* of variable $i$. Define the *projection graph* $G_\pi = (V_\pi, E_\pi)$ so that there is a node for each $\pi_i^q \in \pi_\mathbf{i}$, and nodes $\pi_i^q$ and $\pi_j^r$ are connected if $(i,j) \in E$. We call the graph consisting of all possible variable partitions the *full projection graph*. In Figure 1 we show part of the full projection graph corresponding to one edge $(i,j)$, where $x_i$ has three values and $x_j$ has four values. Intuitively, a partition for a variable splits its values into two clusters, resulting in a binary variable. For example, the (new) variable corresponding to the partition $\{0,1\}\{2\}$ of $x_i$ is 1 if $x_i = 2$, and 0 otherwise. The following gives a projection of marginal vectors of non-binary MRFs onto the marginal polytope of the projection graph $G_\pi$, which has binary variables for each partition.

**Definition 1.** The linear map $\Psi_\pi$ takes $\mu \in \mathcal{M}$ and for each node $v = \pi_i^q \in V_\pi$ assigns $\mu'_{v;1} = \sum_{s \in \chi_i \text{ s.t. } \pi_i^q(s)=1} \mu_{i;s}$ and for each edge $e = (\pi_i^q, \pi_j^r) \in E_\pi$ assigns $\mu'_{e;11} = \sum_{s_i \in \chi_i, s_j \in \chi_j \text{ s.t. } \pi_i^q(s_i)=\pi_j^r(s_j)=1} \mu_{ij;s_i s_j}$.

To construct valid inequalities for each projection we need to characterize the image space. Let $\mathcal{M}_{\{0,1\}}(G_\pi)$ denote the binary marginal polytope of the projection graph.

**Theorem 1.** *The image of the projection $\Psi_\pi$ is $\mathcal{M}_{\{0,1\}}(G_\pi)$, i.e. $\Psi_\pi : \mathcal{M} \to \mathcal{M}_{\{0,1\}}(G_\pi)$.*

*Proof.* Since $\Psi_\pi$ is a linear map, it suffices to show that, for every extreme point $\mu \in \mathcal{M}$, $\Psi_\pi(\mu) \in \mathcal{M}_{\{0,1\}}(G_\pi)$. The extreme points of $\mathcal{M}$ correspond one-to-one with assignments $\mathbf{x} \in \chi^n$. Given an extreme point $\mu \in \mathcal{M}$ and variable $v = \pi_i^q \in V_\pi$, define $\mathbf{x}'(\mu)_v = \sum_{s \in \chi_i \text{ s.t. } \pi_i^q(s)=1} \mu_{i;s}$. Since $\mu$ is an extreme point, $\mu_{i;s} = 1$ for exactly one value $s$, which implies that $\mathbf{x}'(\mu) \in \{0,1\}^{|V_\pi|}$. Then, $\Psi_\pi(\mu) = E[\phi(\mathbf{x}'(\mu))]$, showing that $\Psi_\pi(\mu) \in \mathcal{M}_{\{0,1\}}(G_\pi)$. $\qquad\square$

This result allows valid inequalities for $\mathcal{M}_{\{0,1\}}(G_\pi)$ to carry over to $\mathcal{M}$. In general, the projection $\Psi_\pi$ will not be surjective. Suppose every variable has $k$ values. The *single projection graph*,

where $|\pi_{\mathbf{i}}| = 1$ for all $i$, has one node per variable and *is* surjective. The full projection graph has $O(2^k)$ nodes per variable. A cutting-plane algorithm may begin by projecting onto a small graph, then expanding to larger graphs only after satisfying all inequalities given by the smaller one. The $k-$*projection graph* $G_k = (V_k, E_k)$ has $k$ partitions per variable corresponding to each value versus all the other values.

These projections yield a new class of cycle inequalities for the marginal polytope. Consider a single projection graph $G_\pi$, a cycle $C$ in $G$, and any $F \subseteq C$ such that $|F|$ is odd. Let $\pi_i$ be the partition for node $i$. We obtain the following valid inequality for $\mu \in \mathcal{M}$ by applying the projection $\Psi_\pi$ and the cycle inequality:

$$\sum_{(i,j) \in C \setminus F} \mu_{ij}^\pi (x_i' \neq x_j') + \sum_{(i,j) \in F} \mu_{ij}^\pi (x_i' = x_j') \geq 1, \tag{5}$$

where

$$\mu_{ij}^\pi (x_i' \neq x_j') = \sum_{s_i \in \chi_i, s_j \in \chi_j \text{ s.t. } \pi_i(s_i) \neq \pi_j(s_j)} \mu_{ij;s_i s_j} \tag{6}$$

$$\mu_{ij}^\pi (x_i' = x_j') = \sum_{s_i \in \chi_i, s_j \in \chi_j \text{ s.t. } \pi_i(s_i) = \pi_j(s_j)} \mu_{ij;s_i s_j}. \tag{7}$$

It is revealing to contrast (5) with $\sum_{(i,j) \in C \setminus F} \delta(x_i \neq x_j) + \sum_{(i,j) \in F} \delta(x_i = x_j) \geq 1$. For $\mathbf{x} \in \chi^n$, the latter holds only for $|F| = 1$. We can only obtain the more general inequality by fixing a partition of each node's values.

**Theorem 2.** *For every single projection graph $G_\pi$ and every cycle inequality arising from a chordless circuit $C$ on $G_\pi$, $\exists \mu \in \text{LOCAL}(G) \setminus \mathcal{M}$ such that $\mu$ violates that inequality.*

*Proof.* For each variable $i \in V$, choose $s_i, t_i$ s.t. $\pi_i(s_i) = 1$ and $\pi_i(t_i) = 0$. Assign $\mu_{i;q} = 0$ for $q \in \chi_i \setminus \{s_i, t_i\}$. Similarly, for every $(i,j) \in E$, assign $\mu_{ij;qr} = 0$ for $q \in \chi_i \setminus \{s_i, t_i\}$ and $r \in \chi_j \setminus \{s_j, t_j\}$. The polytope resulting from the projection of $\mathcal{M}$ onto the remaining values (e.g. $\mu_{i;s_i}$) is isomorphic to $\mathcal{M}_{\{0,1\}}$ for the graph $G_\pi$. Barahona and Mahjoub [4] showed that the cycle inequality on the chordless circuit $C$ is facet-defining for $\mathcal{M}_{\{0,1\}}$. Since $C$ is over $\geq 3$ variables from $G$, this cannot be a facet of $\text{LOCAL}(G)$. Let $\text{LOCAL}_{\{0,1\}}$ be the projection of $\text{LOCAL}(G)$ onto the remaining values. Thus, $\exists \mu' \in \text{LOCAL}_{\{0,1\}} \setminus \mathcal{M}_{\{0,1\}}$, and we can assign $\mu$ accordingly. $\square$

Note that the theorem implies that the projected cycle inequalities are strictly tighter than $\text{LOCAL}(G)$, but it does not characterize how much is gained.

If all $n$ variables have $k$ values, then there are $O((2^k)^n)$ different single projection graphs. However, since for every cycle inequality in the single projection graphs there is an equivalent cycle inequality in the full projection graph, it suffices to consider just the full projection graph. Thus, even though the projection is not surjective, the full projection graph, which has $O(n2^k)$ nodes, allows us to efficiently obtain a tighter relaxation than any combination of projection graphs would give. In particular, the separation problem for all cycle inequalities (5) for all single projection graphs, when we allow some additional valid inequalities for $\mathcal{M}$ (arising from the cycle using more than one partition for some variables), can now be solved in time $O(poly(n, 2^k))$.

**Related work.** In earlier work, Althaus et al. [1] analyze the *GMEC polyhedron*, which is equivalent to the marginal polytope. They use a similar value-aggregation technique to derive valid constraints from the triangle inequalities. Koster et al. [8] investigate the *Partial Constraint Satisfaction Problem polytope*, which is also equivalent to the marginal polytope. They used value-aggregation to show that a class of cycle inequalities (corresponding to Eq. 5 for $|F| = 1$) are valid for this polytope, and give an algorithm to separate the inequalities for a single cycle. Interestingly, both papers showed that these constraints are facet-defining.

**Non-pairwise Markov random fields.** These results could be applied to non-pairwise MRFs by first projecting the marginal vector onto the marginal polytope of a pairwise MRF. More generally, suppose we include additional variables corresponding to the joint probability of a cluster of variables. We need to add constraints enforcing that all variables in common between two clusters have the same marginals. For pairwise clusters these are simply the usual local consistency constraints. We can now apply the projections of the previous section, considering various partitions of each cluster variable, to obtain a tighter relaxation of the marginal polytope.

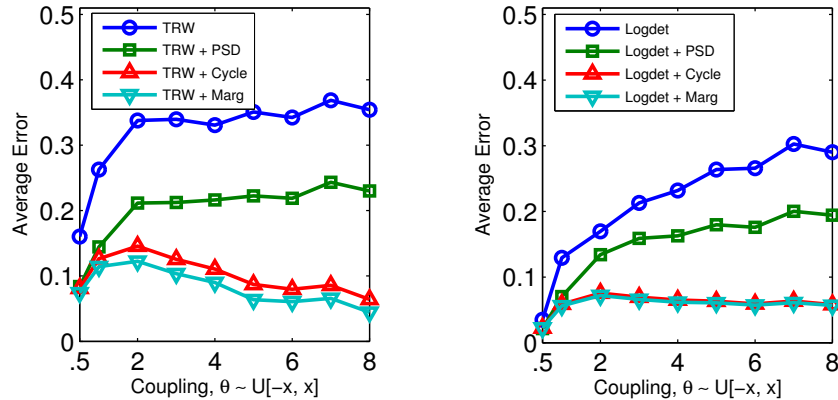

Figure 2: Accuracy of single node marginals on 10 node complete graph (100 trials).

## 5    Experiments

**Computing marginals**. We experimented with Algorithm 1 using both the log-determinant [12] and the TRW [10] entropy approximations. These trials are on *Ising models*, which are pairwise MRFs with $x_i \in \{-1, 1\}$ and potentials $\phi_i(\mathbf{x}) = x_i$ for $i \in V$ and $\phi_{ij}(\mathbf{x}) = x_i x_j$ for $(i, j) \in E$. Although TRW can efficiently optimize over the spanning tree polytope, for these experiments we simply use a weighted distribution over spanning trees, where each tree's weight is the sum of the absolute value of its edge weights $\theta_{ij}$. The edge appearance probabilities for this distribution can be efficiently computed using the Matrix Tree Theorem [13]. We optimize the TRW objective with conditional gradient, using linear programming after each gradient step to project onto OUTER. We used the glpkmex and YALMIP optimization packages within Matlab, and wrote the separation algorithm for the cycle inequalities in Java.

In Figure 2 we show results for 10 node complete graphs with $\theta_i \sim U[-1, 1]$ and $\theta_{ij} \sim U[-x, x]$, where the coupling strength is varied along the $x$-axis of the figure. For each data point we averaged the results over 100 trials. The $y$-axis shows the average $\ell_1$ error of the single node marginals. These MRFs are highly coupled, and loopy belief propagation (not shown) with a .5 decay rate seldom converges. The TRW and log-determinant algorithms, optimizing over the local consistency polytope, give pseudomarginals only slightly better than loopy BP. Even adding the positive semi-definite constraint $M_1(\mu) \succeq 0$, for which TRW must be optimized using conditional gradient and semidefinite programming for the projection step, does not improve the accuracy by much. However, both entropy approximations give significantly better pseudomarginals when used by our algorithm together with the cycle inequalities (see "TRW + Cycle" and "Logdet + Cycle" in the figures). For small MRFs, we can exactly represent the marginal polytope as the convex hull of its $2^n$ vertices. We found that the cycle inequalities give nearly as good accuracy as the exact marginal polytope (see "TRW + Marg" and "Logdet + Marg").

Our work sheds some light on the relative value of the entropy approximation compared to the relaxation of the marginal polytope. When the MRF is weakly coupled, both entropy approximations do reasonably well using the local consistency polytope. This is not surprising: the limit of weak coupling is a fully disconnected graph, for which both the entropy approximation and the marginal polytope relaxation are exact. With the local consistency polytope, both entropy approximations get steadily worse as the coupling increases. In contrast, using the exact marginal polytope, we see a peak at $\theta = 2$, then a steady improvement in accuracy as the coupling term grows. This occurs because the limit of strong coupling is the MAP problem, for which using the exact marginal polytope will give exact results. The interesting region is near the peak $\theta = 2$, where the entropy term is neither exact nor outweighed by the coupling. Our algorithm seems to "solve" the part of the problem caused by the local consistency polytope relaxation: TRW's accuracy goes from .33 to .15, and log-determinant's accuracy from .17 to .076. The fact that neither entropy approximation can achieve accuracy below .07, even with the exact marginal polytope, motivates further research on improving this part of the approximation.

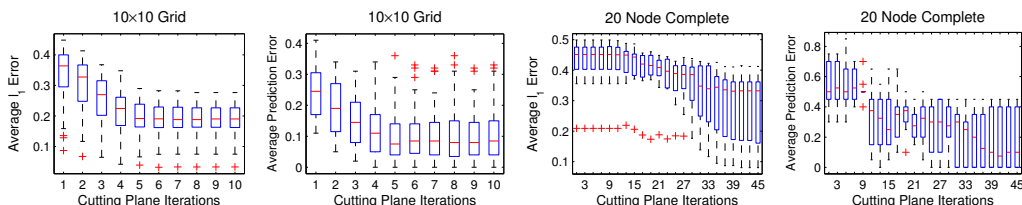

Figure 3: Accuracy of single node marginals with TRW entropy, $\theta_i \in U[-1, 1]$ and $\theta_{ij} \in U[-4, 4]$.

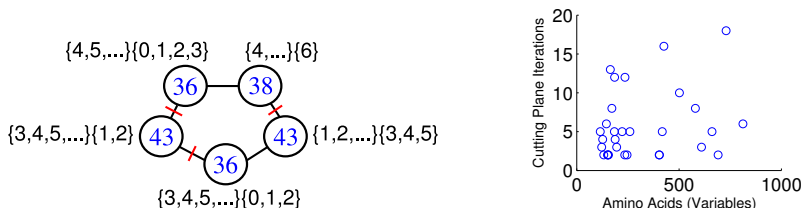

Figure 4: MAP for protein side-chain prediction with Rosetta energy function.

Next, we looked at the number of iterations (in terms of the loop in Algorithm 1) the algorithm takes before all cycle inequalities are satisfied. In each iteration we add to OUTER at most[2] $n$ violated cycle inequalities, coming from the $n$ shortest paths. In Figure 3 we show boxplots of the $l_1$ error of the single node marginals for both 10x10 grid MRFs (40 trials) and 20 node complete MRFs (10 trials). We also show whether the pseudomarginals are on the correct side of .5, which is important if we were doing prediction based on the results from approximate inference. The middle line gives the median, the boxes show the upper and lower quartiles, and the whiskers show the extent of the data. Iteration 1 corresponds to TRW with only the local consistency constraints. For the grid MRFs, all of the cycle inequalities were satisfied within 10 iterations. We observed the same convergence results on a 30x30 grid, although we could not assess the accuracy due to the difficulty of exact marginals calculation. For the complete graph MRFs, the algorithm took many more iterations before all cycle inequalities were satisfied.

**Protein side-chain prediction**. We next applied our algorithm to the problem of predicting protein side-chain configurations. Given the 3-dimensional structure of a protein's backbone, the task is to predict the relative angle of each amino acid's side-chain. The angles are discretized into at most 45 values. Yanover et al. [14] showed that minimization of the Rosetta energy function corresponds to finding the MAP assignment of a non-binary pairwise MRF. They also showed that the tree-reweighted max-product algorithm [9] can be used to solve the LP relaxation given by LOCAL(G), and that this succeeds in finding the MAP assignment for 339 of the 369 proteins in their data set. However, the optimal solution to the LP relaxation for the remaining 30 proteins, arguably the most difficult of the proteins, is fractional.

Using the $k$-projection graph and projected cycle inequalities, we succeeded in finding the MAP assignment for all proteins except for the protein '1rl6'. We show in Figure 4 the number of cutting-plane iterations needed for each of the 30 proteins. In each iteration, we solve the LP relaxation, and, if the solution is not integral, run the separation algorithm to find violated inequalities. For the protein '1rl6', after 12 cutting-plane iterations, the solution was not integral, and we could not find any violated cycle inequalities using the $k$-projection graph. We then tried using the full projection graph, and found the MAP after just one (additional) iteration. Figure 4 shows one of the cycle inequalities (5) in the full projection graph that was found to be violated. The cut edges indicate the 3 edges in $F$. The violating $\mu$ had $\mu_{36;s} = .1667$ for $s \in \{0, 1, 2, 3, 4, 5\}$, $\mu_{38;6} = .3333$, $\mu_{38;4} = .6667$, $\mu_{43;s} = .1667$ for $s \in \{1, 2, 4, 5\}$, $\mu_{43;3} = .3333$, and zero for all other values of these variables. This example shows that the relaxation given by the full projection graph is strictly tighter than that of the $k$-projection graph.

The commercial linear programming solver CPLEX 10.0 solves each LP relaxation in under 75 seconds. Using simple heuristics, the separation algorithm runs in seconds, and we find each protein's MAP assignment in under 11.3 minutes. Kingsford et al. [7] found, and we also observed, that CPLEX's branch-and-cut algorithm for solving integer linear programs also works well for these problems. One interesting future direction would be to combine the two approaches, using our new outer bounds within the branch-and-cut scheme. Our results show that the new outer bounds are powerful, allowing us to find the MAP solution for all of the MRFs, and suggesting that using them will also lead to significantly more accurate marginals for non-binary MRFs.

## 6  Conclusion

The facial structure of the cut polytope, equivalently, the binary marginal polytope, has been well-studied over the last twenty years. The cycle inequalities are just one of many large classes of valid inequalities for the cut polytope for which efficient separation algorithms are known. Our theoretical results can be used to derive outer bounds for the marginal polytope from any of the valid inequalities on the cut polytope. Our approach is particularly valuable because it takes advantage of the sparsity of the graph, and only uses additional constraints when they are guaranteed to affect the solution. An interesting open problem is to develop new message-passing algorithms which can incorporate cycle and other inequalities, to efficiently do the optimization within the cutting-plane algorithm.

**Acknowledgments**

The authors thank Amir Globerson and David Karger for helpful discussions. This work was supported in part by the DARPA Transfer Learning program. D.S. was also supported by a National Science Foundation Graduate Research Fellowship.

## Footnotes

[1]For reasons of clarity, our results will be given in terms of the binary marginal polytope, also called the *correlation polytope*, which is equivalent to the cut polytope of the suspension graph of the MRF [6].

[2]Many fewer inequalities were added, since not all cycles in $G'$ are simple cycles in $G$.

## References

[1] E. Althaus, O. Kohlbacher, H.-P. Lenhof, and P. Müller. A combinatorial approach to protein docking with flexible side-chains. In *RECOMB '00*, pages 15–24, 2000.

[2] F. Barahona. On cuts and matchings in planar graphs. *Mathematical Programming*, 60:53–68, 1993.

[3] F. Barahona, M. Grötschel, M. Junger, and G. Reinelt. An application of combinatorial optimization to statistical physics and circuit layout design. *Operations Research*, 36(3):493–513, 1988.

[4] F. Barahona and A. R. Mahjoub. On the cut polytope. *Mathematical Programming*, 36:157–173, 1986.

[5] T. H. Cormen, C. E. Leiserson, R. L. Rivest, and C. Stein. *Introduction to Algorithms*. MIT Press, 2nd edition, 2001.

[6] M. M. Deza and M. Laurent. *Geometry of Cuts and Metrics*, volume 15 of *Algorithms and Combinatorics*. Springer, 1997.

[7] C. L. Kingsford, B. Chazelle, and M. Singh. Solving and analyzing side-chain positioning problems using linear and integer programming. *Bioinformatics*, 21(7):1028–1039, 2005.

[8] A. Koster, S. van Hoesel, and A. Kolen. The partial constraint satisfaction problem: Facets and lifting theorems. *Operations Research Letters*, 23:89–97, 1998.

[9] M. Wainwright, T. Jaakkola, and A. Willsky. MAP estimation via agreement on trees: message-passing and linear programming. *IEEE Transactions on Information Theory*, 51(11):3697–3717, November 2005.

[10] M. Wainwright, T. Jaakkola, and A. Willsky. A new class of upper bounds on the log partition function. *IEEE Transactions on Information Theory*, 51:2313–2335, July 2005.

[11] M. Wainwright and M. I. Jordan. Graphical models, exponential families and variational inference. Technical Report 649, UC Berkeley, Dept. of Statistics, 2003.

[12] M. Wainwright and M. I. Jordan. Log-determinant relaxation for approximate inference in discrete Markov random fields. *IEEE Transactions on Signal Processing*, 54(6):2099–2109, June 2006.

[13] D. B. West. *Introduction to Graph Theory*. Prentice Hall, 2001.

[14] C. Yanover, T. Meltzer, and Y. Weiss. Linear programming relaxations and belief propagation – an empirical study. *JMLR Special Issue on Machine Learning and Large Scale Optimization*, 7:1887–1907, September 2006.

[15] J. Yedidia, W. Freeman, and Y. Weiss. Bethe free energy, Kikuchi approximations, and belief propagation algorithms. Technical Report 16, Mitsubishi Electric Research Lab, 2001.

